# Parameterized Novelty Detection for Environmental Sensor Monitoring

**Cynthia Archer, Todd K. Leen, Antonio Baptista**
OGI School of Science & Engineering
Oregon Health & Science University
20000 N. W. Walker Road
Beaverton, OR 97006
archer@cse.ogi.edu, tleen@cse.ogi.edu, baptista@ccalmr.ogi.edu

## Abstract

As part of an environmental observation and forecasting system, sensors deployed in the Columbia RIver Estuary (CORIE) gather information on physical dynamics and changes in estuary habitat. Of these, salinity sensors are particularly susceptible to bio-fouling, which gradually degrades sensor response and corrupts critical data. Automatic fault detectors have the capability to identify bio-fouling early and minimize data loss. Complicating the development of discriminatory classifiers is the scarcity of bio-fouling onset examples and the variability of the bio-fouling signature. To solve these problems, we take a novelty detection approach that incorporates a parameterized bio-fouling model. These detectors identify the occurrence of bio-fouling, and its onset time as reliably as human experts. Real-time detectors installed during the summer of 2001 produced no false alarms, yet detected all episodes of sensor degradation before the field staff scheduled these sensors for cleaning. From this initial deployment through February 2003, our bio-fouling detectors have essentially *doubled* the amount of useful data coming from the CORIE sensors.

## 1 Introduction

Environmental observation and forecasting systems (EOFS) gather, process, and deliver environmental information to facilitate sustainable development of natural resources. Our work is part of a pilot EOFS system being developed for the Columbia River Estuary (CORIE) [1]. This system uses data from sensors deployed throughout the estuary (Figure 1) to calibrate and verify numerical models of circulation and material transport. CORIE scientists use these models to predict and evaluate the effects of development on the estuary environment (e.g. [2]).

CORIE salinity sensors deployed in the estuary lose several months of data every year due to sensor degradation. Corrupted and missing field measurements compromise model calibration and verification, which can lead to invalid environmental forecasts. The most common form of salinity sensor degradation is *bio-fouling*, a

reduction of the sensor response due to growth of biological material on the sensor.

Prior the deployment of the technology described here, on a yearly basis CORIE salinity sensors suffered a 68% data loss due to bio-fouling. Although bio-fouling degradation is a common problem for environmental sensors, there is apparently no previous work that develops automatic detectors of such degradation.

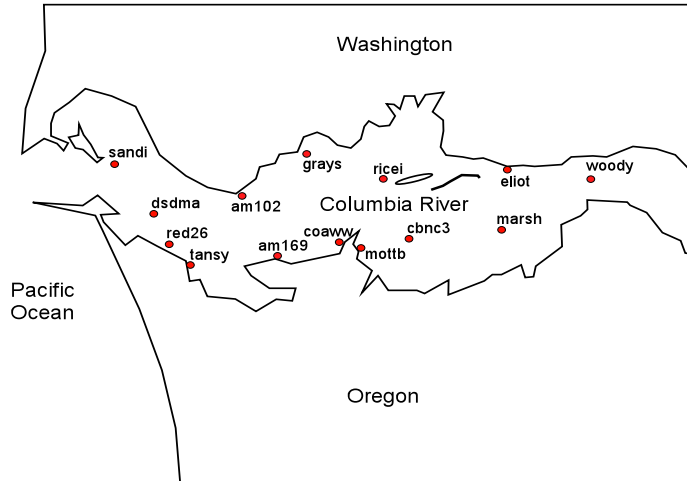

Figure 1: Map of Columbia River estuary marked with locations of CORIE sensors.

Early bio-fouling detection is made difficult by the normal variability of salinity measurements. Tides cause the measurements to vary from near river salinity to near ocean salinity twice a day. The temporal pattern of salinity penetration varies spatially in the estuary. In addition, upriver sites, such as AM169, show substantial variability with the 14 and 28 day spring-neap tidal cycle. Changes in weather (e.g. winds, precipitation) and ocean conditions cause additional variations in salinity.

To complicate bio-fouling detection further, the bio-fouling signature also varies from episode to episode. The time from onset to complete bio-fouling can take anywhere from 3 weeks to 5 months depending on the season and type of growth. We observe two types of bio-fouling in the estuary, hard growth (e.g. barnacles) characterized by *quick* linear degradation and soft growth (e.g. plant material) characterized by *slow* linear degradation with occasional interruptions in the down-trend.

Figure 2 illustrates tidal variations in salinity and the effect that bio-fouling has on these measurements. It contains salinity time series in practical salinity units (psu) from two sensors mounted at the Red26 station, Figure 1. The upper trace, from sensor CT1460, contains only clean measurements. The lower trace, from sensor CT1448, contains both clean and bio-fouled measurements. The first half of the two time series are similar, but beginning on September $28^{th}$, the salinity measurements diverge. The CT1448 sensor exhibits typical hard-growth bio-fouling degradation.

The primary challenge to our work is to detect the degradation *quickly*, ideally within several diurnal cycles. Early detection will limit the use of corrupted data in on-line applications, and provide a basis to rapidly replace degrading sensors, and thus drastically reduce data loss.

Although the CORIE data archives contain many months of bio-fouled data, there are relatively *few* examples of the *onset* of degradation for most of the sensors

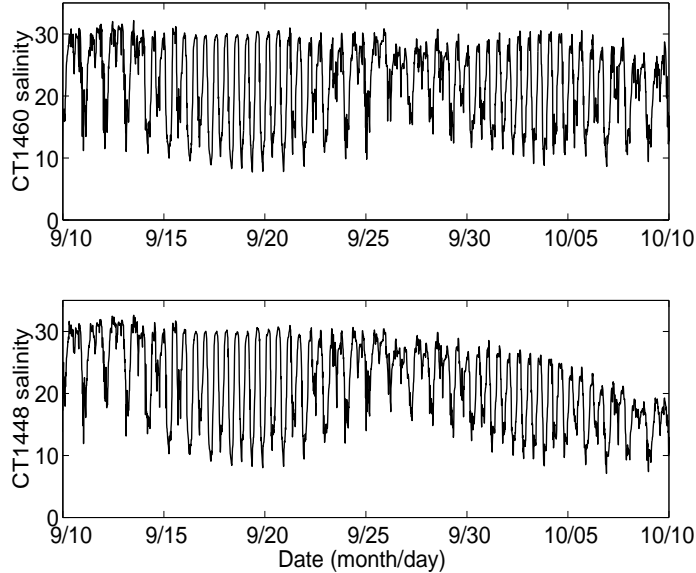

Figure 2: Clean and bio-fouled salinity time series examples from Red26 station. The upper time series is from clean instrument CT1460. The lower time series from instrument CT1448 shows degradation beginning on September 28, 2001. On removal, CT1448 was found to be bio-fouled.

deployed in the estuary, and it is this *onset* that we must detect. The dearth of onset examples, and the observed variability of the bio-fouling signature spatially, seasonally, and weekly (according to the spring/neap tidal cycle) prevents use of classical discriminatory fault detectors. Instead we develop a *parameterized novelty detector* to detect bio-fouling. This detector incorporates a parameterized model of bio-fouling behavior. The parameters in the model of bio-fouled sensor behavior are fit *on-line* by maximum-likelihood estimation. A model of the clean sensor behavior is fit to archival data. These models are used in a sequential likelihood test to provide detection of bio-fouling, *and* an estimation of the time at which the degradation began.

Evaluations show that our detectors identify the onset of bio-fouling as reliably as human experts, and frequently within fewer tidal cycles of the onset. Our deployment of sensors throughout the estuary has resulted in an actual reduction of the error loss from 68% to 35%. However, this figure does not adequately reflect the efficacy of the detectors. Were it economical to replace sensors *immediately* upon detection of degradation, the data loss would have been reduced to 17%.

## 2 Salinity and Temperature

Our detectors monitor maximum diurnal (md) salinity, defined as the maximum salinity near one of the two diurnal tidal floods. When the sensor is clean, the md salinity stays close to some mean value, with occasional dips of several psu caused by variations in the intrusion of salt water into the estuary. When the sensor bio-fouls, the md salinity gradually decreases to typically less than half its normal mean value, as seen in the Figure 2 example.

Detectors that monitor salinity alone can not distinguish between normal decreases

in salinity and early bio-fouling. This results in a high false alarm rate[1]. Natural salinity decreases can be recognized by monitoring a correlated source of information that is *not* corrupted by bio-fouling.

Salinity and temperature at a station are products of the same mixing process of ocean and river waters, so we expect these values will be correlated. Assuming linear mixing of ocean and river waters, measured salinity $S_m$ and temperature $T_m$ are linear functions of ocean $\{S_o, T_o\}$ and river $\{S_r, T_r\}$ values

$$S_m = \alpha(t)S_o + (1 - \alpha(t))S_r \tag{1}$$

$$T_m = \alpha(t)T_o + (1 - \alpha(t))T_r \tag{2}$$

where $\alpha(t)$ is the mixing coefficient at time $t$. River salinity $S_r$ is close to zero. Consequently, the estimated mixing coefficient

$$\alpha(t) = \frac{T_r - T_m}{T_r - T_o} \tag{3}$$

should be well correlated with salinity, $S_m \approx \alpha S_o$. The river temperature is measured at far upstream stations (Elliot or Woody). The ocean temperature is estimated from measurements at Sand Island, the outermost sensor station.

## 3  Bio-fouling Detection

Our early experiments with single-measurement detection suggested that we develop detectors that accrue information over time - similar to the standard sequential likelihood methods in classical pattern recognition. The is a natural framework for detecting degradation that grows with time.

Assume a sequence of measurements (salinity and temperature) $y_n$, $n = 1, \ldots, N$ where $N$ is the current time. We construct probability densities for such sequences for both *clean* sensors $p(y_1, \ldots, y_N \mid c)$, and for biofouled sensors $p(y_1, \ldots, y_N \mid f)$. With these distributions, we construct a likelihood ratio test

$$h \;=\; \ln \frac{p(y_1, \ldots, y_N \mid f)}{p(y_1, \ldots, y_N \mid c)} \; \overset{f}{\underset{c}{\gtrless}} \; \lambda \tag{4}$$

where the threshold $\lambda$ is chosen high enough to provide a specified false alarm rate (Neyman-Pearson test).

We assume that the probability density for the measurement sequence for *fouled* detectors is parameterized by a vector of *unknown* parameters $\theta$. The model is constructed such that at $\theta = 0$ the density for the sequence assuming a fouled detector is equal to the density of the sequence assuming a clean detector

$$p(y_1, \ldots, y_N \mid f, \theta = 0) \;=\; p(y_1, \ldots, y_N \mid c) \tag{5}$$

Next, we suppose that a given sequence contains a bio-fouling event that is initiated at the *unknown* time $\tau$. Under our density models (below), consecutive measurements in the sequence are independent *conditioned on the state of the detector*.

Consequently, the likelihood ratio for the sequence (4) reduces to

$$
\begin{aligned}
h &= \ln \frac{p(y_1, \ldots, y_N \mid f, \tau, \theta)}{p(y_1, \ldots, y_N \mid c)} = \ln \frac{p(y_1, \ldots, y_{\tau-1} \mid c)\, p(y_\tau, \ldots, y_N \mid \tau, \theta, f)}{p(y_1, \ldots, y_N \mid c)} \\
&= \sum_{n=\tau}^{N} \ln \frac{p(y_n \mid \tau, \theta, f)}{p(y_n \mid c)} \underset{c}{\overset{f}{\underset{<}{\gtrless}}} \lambda
\end{aligned}
\tag{6}
$$

Finally, we fit the fouling model parameters $\theta$ *and the onset time* $\tau$, by maximizing the log-likelihood $\ln p(y_1, \ldots, y_N \mid f, \tau, \theta)$ with respect to $\theta$ and $\tau$. Since the *clean* detector model is independent of $\tau$ and $\theta$, this is equivalent to maximizing the log-likelihood ratio in (6). Hence, we replace the latter with

$$
h = \max_{\tau, \theta} \sum_{n=\tau}^{N} \ln \frac{p(y_n \mid \tau, \theta, f)}{p(y_n \mid c)} \underset{c}{\overset{f}{\underset{<}{\gtrless}}} \lambda
\tag{7}
$$

If the sequence is coming from a *clean* sensor, the fit should give $\theta \approx 0$ and hence $h \approx 0$ (cf 5), and we will detect no event (assuming $\lambda > 0$). This construction is a variant of the type of signal change detection discussed by Basseville [3].

### 3.1 Bio-fouling Fault Model

By parameterizing the bio-fouling model, we are able to develop detectors using only clean example data. In this parameterized novelty detector, the bio-fouled parameters $\theta$ are fit on-line to the data under test. To develop our classifier, we first define models of the clean and bio-fouled data. We model the *true* salinity, $s$, and temperature-based mixing coefficient, $\alpha$, as jointly Gaussian,

$$
p(s, \alpha \mid c) = \mathcal{N}(\mu, \Sigma) \text{ where } \mu = \begin{bmatrix} \mu_s \\ \mu_\alpha \end{bmatrix} \text{ and } \Sigma = \begin{bmatrix} \sigma_s^2 & \sigma_{s\alpha} \\ \sigma_{s\alpha} & \sigma_\alpha^2 \end{bmatrix}.
\tag{8}
$$

This provides a regression of the salinity on $\alpha$. The probability of md salinity *measurement* conditioned on temperature when the sensor is clean is Gaussian $\mathcal{N}(\eta, \rho^2)$, with conditional mean

$$
\mathsf{E}[s \mid \alpha, c] \equiv \eta = \mu_s + (\sigma_{s\alpha}/\sigma_\alpha^2)(\alpha - \mu_\alpha)
\tag{9}
$$

and conditional variance

$$
\mathrm{var}[s \mid \alpha, c] \equiv \rho^2 = \sigma_s^2 - \sigma_{s\alpha}^2/\sigma_\alpha^2
\tag{10}
$$

When bio-fouling occurs, the salinity measurement is suppressed relative to the true value. We model this suppression as a linear downtrend with (unknown) rate (slope) $m$ that begins at (unknown) time $\tau$. The model of the measured md salinity value for a fouled detector is

$$
x_n = g(n)s_n
\tag{11}
$$

where the suppression factor, $g(n)$, is

$$
g(n) = \begin{cases} 1 & n < \tau \\ (1 - m(n - \tau)) & n \geq \tau \end{cases}
\tag{12}
$$

and $m$ is the bio-fouling rate (1/sec). Using this suppression factor $g(n)$ (12), the probability of the salinity measurement, $x$, conditioned on temperature is

$$
\mathrm{p}(x_n \mid \alpha_n, m, \tau, f) = \mathcal{N}(g(n)\eta_n, g^2(n)\rho^2)
\tag{13}
$$

Note that since the temperature sensor is not susceptible to bio-fouling, we need not consider the case of both sensors degrading at the same time.

The discriminant function in (7) depends on the parameters of the clean model (9) and (10) which are estimated from historical data. It also depends on the slope parameter $\theta = m$ of the fouling model, and the onset time $\tau$ which are fit online as per (7).

Applying our Gaussian models in (8) and 13) to (7) gives us

$$h = \max_{\tau, m} \sum_{n=\tau}^{N} \ln \frac{1}{1 - m(n - \tau)} + \frac{(x_n - \eta_n)^2}{2\rho^2} - \frac{(x_n - (1 - m(n - \tau))\eta_n)^2}{2(1 - m(n - \tau))^2 \rho^2} \quad (14)$$

When $h$ is above our chosen threshold, the detector signals a biofouled sensor. The threshold $\lambda$ is set to provide a maximum false alarm rate on historical data.

## 3.2 Model Fitting

We find maximum likelihood estimates for $\mu$ and $\Sigma$ from clean archival time series data. For $y_n = [s_n, \alpha_n]^T$ and $N$ training values, the mean is given by $\mu = \frac{1}{N} \sum_n y_n$ and the covariance matrix by $\Sigma = \frac{1}{N} \sum_n (y_n - \mu)(y_n - \mu)^T$. All other classifier parameter values, such as $\mu_s$ or $\mathsf{E}[s|\alpha]$, can be extracted or calculated from $\mu$ and $\Sigma$.

At each time step $N$, we determine the maximum likelihood estimate of onset time $\tau$ and bio-fouling rate $m$ from the data under test. We find the maximum likelihood estimate of bio-fouling rate $m$, for some onset time $\tau$, by setting the first derivative of (14) with respect to $m$ equal to zero. This operation yields the relation

$$m \sum_{k=\tau+1}^{N} \frac{(k - \tau)^2}{\omega_k^2} \eta_k^2 = \sum_{k=\tau+1}^{N} \frac{k - \tau}{\omega_k} \left( \frac{(x_k - \eta_k)\eta_k}{\omega_k} - \rho^2 + \frac{(x_k - \omega_k \eta_k)^2}{\omega_k^2} \right) \quad (15)$$

where $\omega_k = 1 - m(k - \tau)$ and $N$ is the current time. Note that $m$ appears both at the beginning of (15) and in the definition of $\omega$, so we do not have a closed form solution for $m$. However, the $\omega$ values act as weights that increase the importance of most recent measurements. This weighting accounts for the expected decrease in measurement variance as bio-fouling progresses. To estimate $m$ we take an iterative approach. First, initialize $m$ to its minimum mean-squared error value given by

$$m^{(0)} = -\frac{\sum_{k=\tau+1}^{N} (k - \tau)(x_k - \eta_k)\eta_k}{\sum_{k=\tau+1}^{N} (k - \tau)^2 \eta_k^2} \quad (16)$$

Second, repeatedly solve (15) for $m^{(i)}$ with $\omega$ calculated using the previous value $m^{(i-1)}$. The estimated rate value stops changing when $h$ reaches a maximum.

If we set the window length $N - k$ to maximize the log likelihood ratio, $h$, the best estimate of onset time is $\tau$. To determine the onset time estimate, $\tau$, we search over over all past time for the value of $k$ that maximizes $h$ (14). For each possible window length, that is $k = 3 \ldots N$, we determine the maximum likelihood estimate for $m$ and then calculate the corresponding discriminant $h$. The estimated onset time $\tau$ is the window length $N - k$ that gives the largest value of $h$. If this $h$ is above our threshold, the current measurement is classified as bio-fouled.

## 4 On-line Bio-fouling Detectors

To see how well our classifiers worked in practice, we implemented versions that operated on *real-time* salinity and temperature measurements. For all four instances

of sensor degradation (three bio-fouling incidents and one instrument failure that mimicked bio-fouling) that occurred in the summer 2001 test period, our classifiers correctly indicated a sensor problem before the field staff was aware of it. In addition, the real-time classifiers produced no false alarms during the summer test period. More in-depth discussion of the detector suite is given by Archer et al in [4].

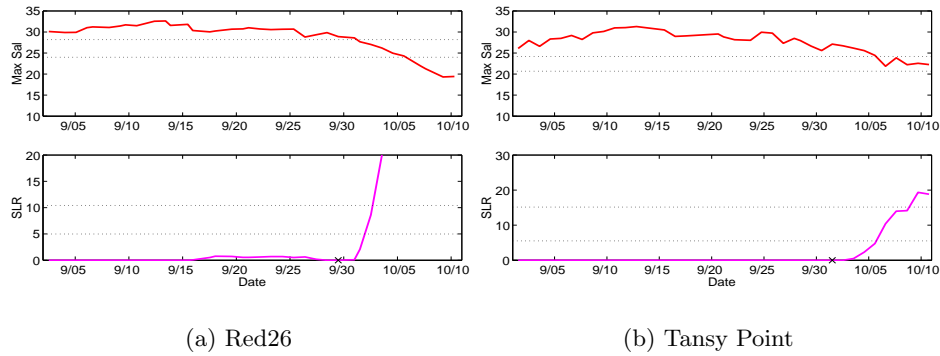

(a) Red26                                    (b) Tansy Point

Figure 3: Bio-fouling Indicators Red26 and Tansy Point. Top plots show maximum diurnal salinity. Dotted lines indicate historical no false alarm (lower) and 10% false alarm rate (upper). Field staff schedule sensors for cleaning when the maximum salinity drops "too low", roughly the no false alarm level. Bottom plots show the sequential likelihood discriminant for forty days of salinity and temperature measurements. Dotted lines indicate historical no false alarm (upper) and 10% false alarm rate (lower). The × indicates the estimated bio-fouling onset time.

The on-line monitor displays a bio-fouling indicator for the previous forty days of data. Figure 3 shows the on-line bio-fouling monitor during incidents at the Red26 CT1448 sensor and the Tansy Point CT1462 sensor. Since we had another sensor mounted at the Red26 site that did not bio-foul, Figure 2, we were able to estimate the bio-fouling time as September $28^{th}$. Our detector discriminant passed the no false alarm threshold five days after onset and roughly three days before the field staff decided the instrument needed cleaning. This reduction in time to detection corresponds to reduced data loss of over 30%. In addition, the onset time estimate of September $29^{th}$ was within a day of the true onset time.

The Tansy Point CT1462 sensor began to bio-foul a few days after the Red26 CT1448 sensor. Our detector indicated that the Tansy Point sensor was bio-fouling on October $9^{th}$. Since neighboring sensor Red26 was being replaced on October $11^{th}$, the field staff decided to retrieve the Tansy Point sensor as well. On removal, this sensor was found to be in the early stages of bio-fouling. In this case, indications from our classifier permitted the sensor to be replaced *before* the field staff would normally have scheduled it for retrieval. Experience with our on-line bio-fouling indicators demonstrates that these methods substantially reduce time from bio-fouling onset to detection.

In addition to the events described above, we have fairly extensive experience with the online detectors since their initial deployment in the Spring of 2001. At this writing we have bio-fouling detectors at all observing stations in the estuary and experience with events throughout the year. Near the end of October, 2001 we experienced a false alarm in a sensor near the surface in the lower estuary. In this case, a steady downward trend in surface salinity, caused by several days of

*rain* triggered a detector response. Following cessation of the precipitation, the discriminant function $h$ returned back to sub-threshold levels.

In a recent (February 2003) study of five sensor stations in the estuary we compared data loss prior to the deployment of bio-fouling detectors, with data loss post-deployment. The pre-deployment period included approximately four years of data from 1997 through the summer of 2001. The post-deployment period ran from spring/summer of 2001 through February 2003.

Neglecting seasonal variation, prior to the deployment of our detectors, 68% of all the sensor data was corrupted by bio-fouling. Following deployment, the rate of data loss due to bio-fouling dropped to 35%. This is the *actual* data loss, and includes delay in responding to the event detection. Were it economical to replace the sensors immediately upon detection of bio-fouling, the data loss rate would have been dropped farther to 17%. Even with the delay in responding to event detection, the detectors have more than *doubled* the amount of reliable data collected from the estuary.

## 5    Discussion

CORIE salinity sensors lose several months of data every year due to sensor bio-fouling. Developing discriminatory fault detectors for these sensors is hampered by the variability of the bio-fouling time-signature, and the dearth of bio-fouling onset example data for training. To solve this problem, we built parameterized novelty detectors. Clean sensor models were developed based on archive data, while bio-fouled sensor models are given a simple parametric form that is fit online. On-line bio-fouling detectors deployed during the summer of 2001 detected all episodes of sensor degradation several days before the field staff without generating any false alarms. Expanded installation of a suite of detectors throughout the estuary continue to successfully detect bio-fouling with minimal false alarm intrusion. The detector deployment has effectively *doubled* the amount of clean data available from the estuary salinity sensors.

**Acknowledgements**

We thank members of the CORIE team, Arun Chawla and Charles Seaton, for their help in acquiring appropriate sensor data, Michael Wilkin for his assistance in labeling the sensor data, and Haiming Zheng for carrying forward the sensor development and deployment and providing the comparison of data loss rates before and after the detector deployment.. This work was supported by the National Science Foundation under grants ECS-9976452 and CCR-0082736.

## Footnotes

[1]Equivalently, if the alarm threshold is increased to maintain a low false alarm rate, the rate of *proper detections* is decreased.

## References

[1] A. Baptista, M. Wilkin, P. Pearson, P. Turner, C. McCandlish, and P. Barrett. Costal and estuarine forecast systems: A multipurpose infrastructure for the Columbia river. *Earth System Monitor*, 9(3), 1999.

[2] U.S. Army Corps of Engineers. Biological asssessment - Columbia river channel improvements project. Technical report, USACE Portland District, December 2001.

[3] M. Basseville. Detecting changes in signals and systems - a survey. *Automatica*, 24(3):309–326, 1988.

[4] C. Archer, A. Baptista, and T.K. Leen. Fault detection for salinity sensors in the Columbia River Estuary. *Water Resources Research*, 39, 2003.
